# How Oscillatory Neuronal Responses Reflect Bistability and Switching of the Hidden Assembly Dynamics

**K. Pawelzik, H.-U. Bauer[†], J. Deppisch, and T. Geisel**
Institut für Theoretische Physik and SFB 185 Nichtlineare Dynamik
Universität Frankfurt, Robert-Mayer-Str. 8-10, D-6000 Frankfurt/M. 11, FRG
[†]temporary adress:CNS-Program, Caltech 216-76, Pasadena
email: klaus@chaos.uni-frankfurt.dbp.de

## Abstract

A switching between apparently coherent (oscillatory) and stochastic episodes of activity has been observed in responses from cat and monkey visual cortex. We describe the dynamics of these phenomena in two parallel approaches, a phenomenological and a rather microscopic one. On the one hand we analyze neuronal responses in terms of a hidden state model (HSM). The parameters of this model are extracted directly from experimental spike trains. They characterize the underlying dynamics as well as the coupling of individual neurons to the network. This phenomenological model thus provides a new framework for the experimental analysis of network dynamics. The application of this method to multi unit activities from the visual cortex of the cat substantiates the existence of oscillatory and stochastic states and quantifies the switching behaviour in the assembly dynamics. On the other hand we start from the single spiking neuron and derive a master equation for the time evolution of the assembly state which we represent by a phase density. This phase density dynamics (PDD) exhibits costability of two attractors, a limit cycle, and a fixed point when synaptic interaction is nonlinear. External fluctuations can switch the bistable system from one state to the other. Finally we show, that the two approaches are mutually consistent and therefore both explain the detailed time structure in the data.

## 1   INTRODUCTION

A few years ago, oscillatory and synchronous neuronal activity was discovered in cat visual cortex [1-3]. These experiments backed earlier considerations about synchrony in neuronal activity as a mechanism to bind features, e.g., of an object in a visual scene [4]. They triggered broad experimental and theoretical investigations of detailed neuronal dynamics as a means for information processing and, in particular, for feature binding. Many theoretical contributions tried to reproduce and explain aspects of the experimentally observed phenomena [5]. Motivated by the experiments, the models where particularly designed to exhibit spatial synchronization of permanent oscillatory responses upon stimulation by a common, connected stimulus like a bar. Most models consist of elements which exhibit a limit cycle after a simple Hopf bifurcation.

The experimental data, however, contain many details which the present models do not yet completely incorporate. One of these details is the coexistence of regular and irregular episodes in the data, which interchange in an apparently stochastic manner. This interchange can be observed in the signals from a single electrode [6] as well as in the time-resolved correlation of the signals from two electrodes [7]. In this contribution we show, that the observed time structure reflects a switching in the dynamics of the underlying neuronal system. This will be demonstrated by two complementary approaches:

On the one hand we present a new method for a quantitative analysis of the dynamical system underlying the measured spike trains. Our approach gives a quantitative description of the dynamical phenomena and furthermore explains the relation between the collective excitation in the network which is not accessible experimentally (i.e. hidden) and the contributions of the single observed neurons in terms of transition probability functions. These probabilities are the parameters of our Ansatz and can be estimated directly from multi unit activities (MUA) using the Baum-Welch-algorithm. Especially for the data from cat visual cortex we find that indeed there are two states dominating the dynamics of collective excitation, namely a state of repeated excitation and a state in which the observed neurons fire independently and stochastically.

On the other hand using simple statistical considerations we derive a description for a local neuronal subpopulation which exhibits bistability. The dynamics of the subpopulation can either rest on a fixed point - corresponding to the irregular firing patterns - or can follow a limit cycle - corresponding to the oscillatory firing patterns. The subpopulation can alternate between both states under the influence of noise in the external excitation. It turns out that the dynamics of this formal model reproduces the observed local cortical signals in much detail.

## 2   Excitability of Neurons and Neuronal Assemblies

An abstract model of a neuron under external excitation $e$ is given by its threshold dynamics. The state of the neuron is represented by its phase $\phi^s$, which is the time passed by since the last action potential ($\phi^s = 0$). The threshold $\Theta$ is high directly after a spike and falls off in time and the neuron can fire again when $e$ exceeds $\Theta$. In case of noise or internal stochasticity, an excitability description of

the dynamics of the neuron is more adequate. It gives the probability $P_f$ to fire again in dependence of the state $\phi^s$ with $P_f(\phi^s) = \sigma(e - \Theta(\phi^s))$ and $\sigma$ some sigmoid function. A monotonously falling threshold $\Theta$ then corresponds to a monotonously increasing excitability $P_f$. Such a description neglects any memory in the neuron going beyond the last spike. In particular this means for an isolated neuron, that $P_f$ can be easily calculated from the inter-spike interval histogram (ISIH) $P_h$ using the relation $P_h(t) = P_f(t) \cdot (1 - \int_o^t P_h(t')dt')$. In that case also the autocorrelation function can be calculated from $P_h(t)$ via $\hat{C}(\tau) = P_h(\tau) + \int_0^\tau P_h(\tau)\hat{C}(\tau - t)dt$.

The excitability formulation sketched above is not valid for a neuron which is embedded in a neuronal assembly. However, we may use this Ansatz of a renewal process to describe the activation dynamics of the whole assembly (see section 5). The phase $\phi^b = 0$ here corresponds to the state of synchronous activity of many neurons in the assembly, which we call *burst* for convenience. Since the dynamics of the network can differ from the dynamics of the elements we expect the function $P_f^b(\phi^b)$ which now describes the burst excitability of the whole assembly to be different from·the spike excitability $P_f(\phi_s)$ of the single neuron.

A simple example for this is a system of integrate and fire neurons in which oscillatory and irregular phases emerge under fixed stimulus conditions([8, 9] and section 5). Contrary to the excitability of the single refractory element the burst excitability $P_f^b$ of the system has a maximum at $\phi^b = T$ which expresses the increased probability to burst again after the typical oscillation period $T$, i.e. the maximum represents a state $o$ of oscillation. The assembly, however, can miss to burst around $\phi^b = T$ with a probability $p_{o \rightarrow s}$ and switch into a second state $s$ in which the probability $p_{s \rightarrow o}$ to burst again is reduced to a constant level. The switching probabilities $p_{o \rightarrow s}$ and $p_{s \rightarrow o}$ can be easily calculated from $P_f^b$. In this way the shape of $P_f^b$ distinguishes a system with an oscillatory state from a system which is purely refractory but which nevertheless can still have strong modulations in the autocorrelogram [13].

## 3  Hidden states and stochastic observables

The single neuron in an assembly, however, need not be strictly coupled to the state of the assembly, i.e. a neuron may also spike for $\phi^b > 0$ and it may not take part in a burst. This stochastic coupling to an underlying process suffices to destroy the equivalence of $P_h$ and the autocorrelogram $C(\tau) = < s(t)s(t + \tau) >_t$ of the spike train $s(t) \in \{0, 1\}$ (Fig 1). We therefore include the probability $P_{obs}(\phi^b)$ to observe a spike when the assembly is in the state $\phi^b$ into our description (Fig. 2). The unlikely case where the spike represents the burst corresponds to the choice $P_{obs} = \delta_{\phi^b,0}$.

## 4  Application to Experimental Data

While our approach is quite general, we here concentrate on the measurements of Gray et al. [2] in cat visual cortex. Because our hidden state model has the structure of a hidden Markov model we can obtain all the parameters $P_{obs}(\phi)$ and

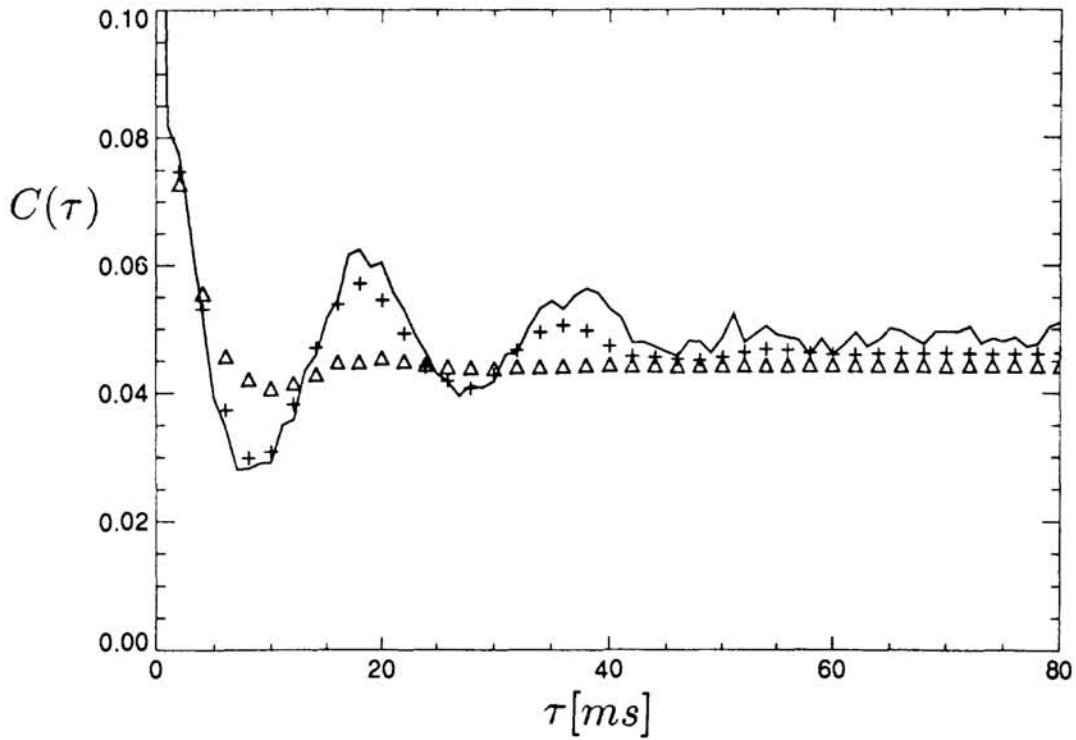

Figure 1: Correlogram of multi unit activities from cat visual cortex (line). Corre-laograms predicted from the ISIH ($\triangle$) and from the hidden state model (+).

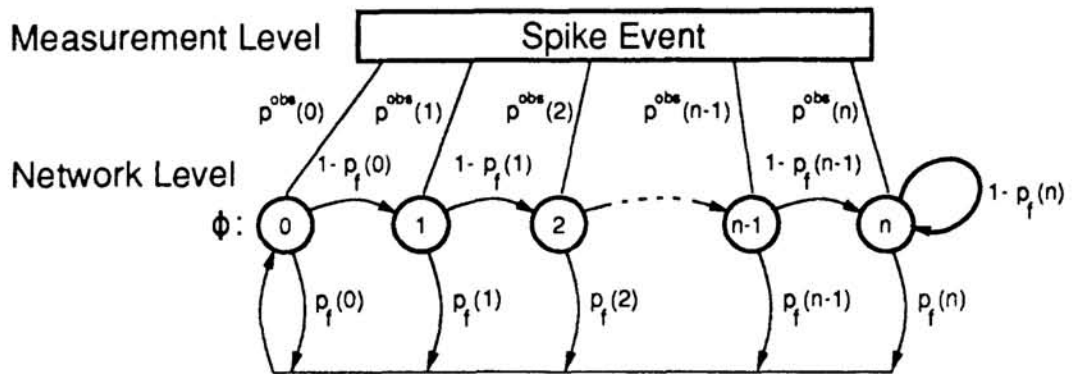

Figure 2: The hidden state model. While $P_f^b(\phi^b)$ governs the dynamics of assembly states $\phi^b$, $P_{obs}(\phi^b)$ represents the probability to observe a spike of a single neuron.

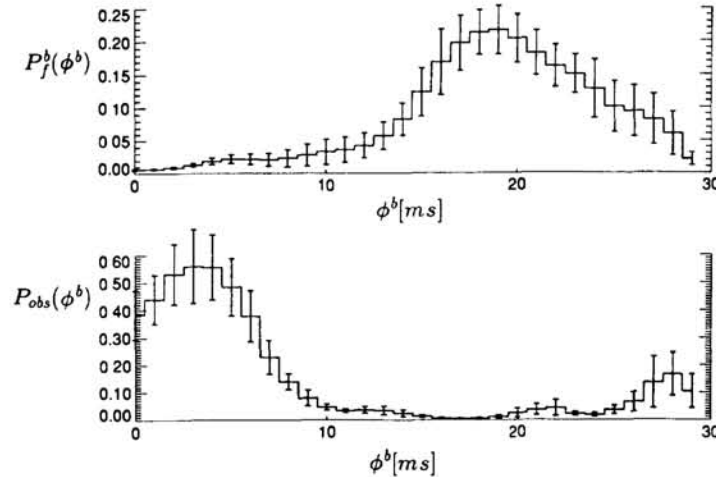

Figure 3: Network excitability $P_f^b$ and single neuron contribution $P_{obs}$ estimated from experimental spike trains (A17, cat).

$P_f^b(\phi)$ directly from the multi unit activities using the well known Baum-Welch algorithm[10]. The results can be seen in Fig. 3. The excitability shows a typical peak around the main period at $T = 19ms$, which indicates a state of oscillation. For larger phases we see a reduced excitability which reveals a state of stochastic activity ($P_{obs}(\phi^b > T) > 0$). The spike observation probability $P_{obs}(\phi)$ is peaked near the burst and is about constant elsewhere. This means that we can characterize the data by a stochastic switching between two dynamical states in the underlying system. Because of the stochastic coupling of the single neuron to the assembly state this can only hardly be observed directly. The switching probabilities between either states calculated from $P_f^b$ coincide with results from other methods [11].

From the excitability $P_f^b$ and the spike probabilities $P_{obs}$ we now obtain the autocorrelation function $\hat{C}(\tau) = \int_\phi \int_{\phi'} P_{obs}(\phi')\mathbf{M}(\phi', \phi)^\tau P_{obs}(\phi)\rho(\phi)d\phi'd\phi$, with $\mathbf{M}$ being the transition matrix of the Markov model (see also below). The result is compared to the true autocorrelation $C(\tau)$ in Fig. 1. The excellent agreement confirms our simple Ansatz of a renewal process for the hidden burst dynamics of the assembly.

## 5   Bistability and Switching in Networks of Spiking Neurons

The above results indicate that the dynamics of a cortical assembly includes bistability rather than a simple Hopf bifurcation. In order to understand how this bistability emerges in a network we go one step back and derive a model for a neuronal subpopulation on the basis of spiking neurons. We assume again that the internal state of the neuron is given by the threshold function $\Theta$ depending on the time since the last spike event and that the excitability of the neuron can be described by a

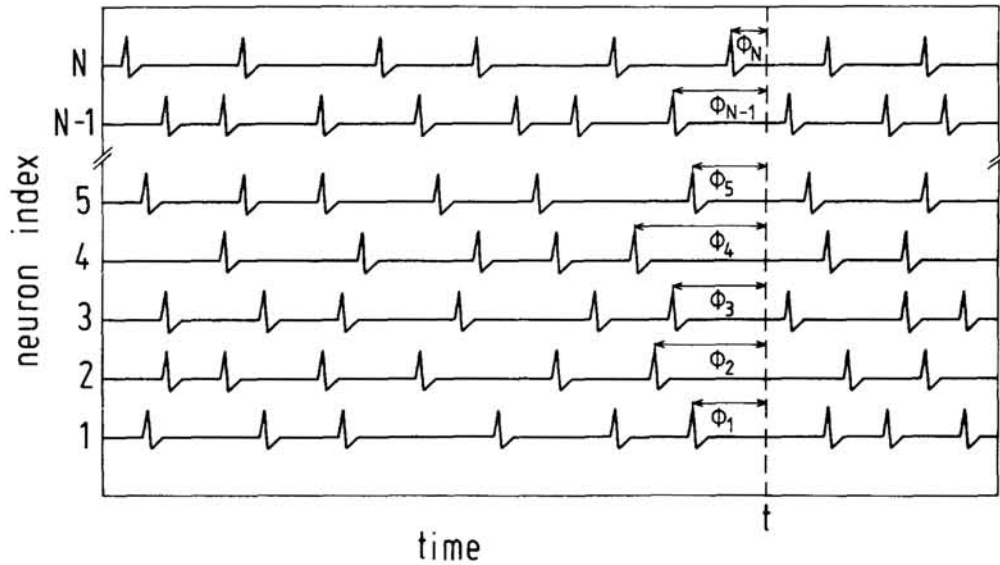

Figure 4: Illustration of assembly state representation by a phase density.

firing probability $P_f$. In a network, however, the input to the neuron has external contributions $i_{ext}$ as well as from within $i_{int}$ i.e.

$$P_f(\phi, t) = \text{sigm}\left(w_{ext} i_{ext}(t) + w_{int} i_{int}(t) - \Theta(\phi)\right).$$

For a more formal treatment of the dynamics of such a network we characterize the assembly state by a phase density $\tilde{\rho}(\phi, t)$ which gives the relative amount of neurons in the assembly which are in phase $\phi$ at time $t$ (Fig 4).

Discretizing the internal phases $\phi$, we transform $\tilde{\rho}(\phi, t)$ to $\vec{\rho}(j)$, a vector whose components $i$ give the probability to be in phase $\phi_i \in [(i-1)\Delta t, i\Delta t]$ at time $t_j = j\Delta t$. The number $T$ of components is chosen large enough to ensure that $p_f(T, j) = P_f(T\Delta t, j\Delta t)$ does not change any more. This vector evolves in time according to

$$\vec{\rho}(j+1) = \mathbf{M}(j)\vec{\rho}(j) \qquad (1)$$

with

$$\mathbf{M}(j) = \begin{pmatrix} 0 & p_f(1,j) & p_f(2,j) & \cdots & p_f(T{-}1,j) & p_f(T,j) \\ 1 & 0 & 0 & & & \\ 0 & 1{-}p_f(1,j) & 0 & \cdots & 0 & \\ & & 1{-}p_f(2,j) & \ddots & \vdots & \\ & & & & 0 & 0 \\ \vdots & & & & & \\ 0 & & \cdots & & 1{-}p_f(T{-}1,j) & 1{-}p_f(T,j) \end{pmatrix},$$

beeing a matrix that incorporates the effects of firing (reset) via the firing probability $p_f(i, j)$.

It remains to define the lateral interaction in the subpopulation. Clearly only the fraction of neurons that fire can interact, therefore we have

$$i_{int} = g(\vec{\rho}_0).$$

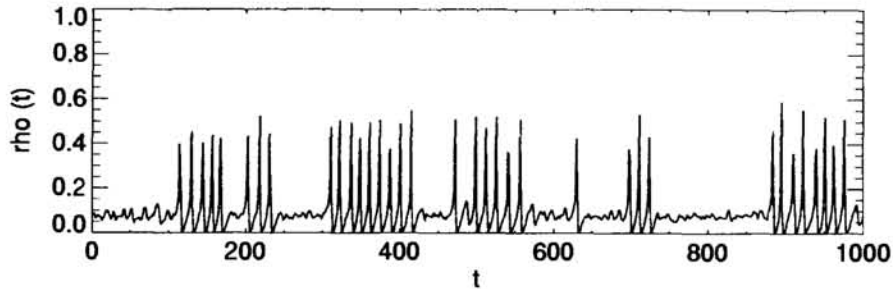

Figure 5: Switching in the assembly dynamics under external fluctuations. Note that $\rho_0$ denotes the time dependent firing rate.

Numerically iterating the dynamics (1) we find, that the distribution can evolve in two distinct ways, depending on the lateral interaction function $g$ and the initialization. First $\vec{\rho}$ can relax to a fixed point, corresponding to a constant fraction of the neurons firing at a particular time. On the level of an individual neuron, this corresponds to a stochastic firing characteristics, and in the measurements this state corresponds to the irregular periods. Secondly the distribution can evolve according to a limit cycle, with a rather large fraction of the neurons in a narrow band of phases i.e. the neurons are synchronous. We find parameter combinations $w_{ext}, w_{int}$ where both states coexist, i.e. we find bistability, if the interaction function is nonlinear, like $g(x) \propto x^2$ or $g(x) \propto x^4$. Nonlinearities of this kind can be brought about by a different type of preferred synapses with a nonlinear transmission characteristic (like an NMDA-receptor synapse) for the corticocortical projections, as compared to the thalamocortical projections. At this point we only want to make the point that our simple model system can exhibit bistability under reasonable assumptions. In the bistable regime some initializations of $\vec{\rho}$ lead to the oscillatory state, some to the stochastic state. Adding some noise to the external input the system can also switch between the two states(Fig. 5). In this way our simple local model can capture the switching phenomenon which is inherent in the experimental data [12].

## 6  Summary and Synthesis

We presented two complementary approaches to the dynamics of neuronal subpopulations, a phenomenological one which captures the time structure in measured spike trains and a neuronal one which provides a formal description of the dynamics of assemblies of spiking neurons. In the phenomenological approach we introduced the hidden state model which revealed that the system underlying the multi unit activities from the cat switches between states of oscillatory and stochastic activity. The analysis of the phase density dynamics showed, that bistability and switching emerges in networks of spiking neurons when the neuronal interaction is nonlinear. It remains to show that these approaches are also quantitatively consistent, i.e. that they are two sides of the same medal. Instead of a formal proof we only remark here that the parameters of the HSM can be extracted directly from the dynamics of the PDD under external noise. For this purpose one only needs to evaluate the interburst interval distribution from which $P_f^b$ can be calculated. $P_{obs}$ is easily estimated as the average shape of $\rho_0(t)$ between successive bursts. We find, that already this procedure gives HSMs which accurately reproduce the correlation function of the

full firing dynamics $\rho_0(t)$. This means that the HSM captures relevant aspects of the assembly dynamics including the relation between the network dynamics and the contributions of single neurons.

## References

[1] Gray C.M., Singer W., *Stimulus-Specific Neuronal Oscillations in Cat Visual Cortex: a Cortical Functional Unit*, Soc.Neurosci.Abstr. 13.404.3 (1989).

[2] Gray C.M., König P., Engel A.K., and Singer W., *Oscillatory Responses in Cat Visual Cortex Exhibit Inter-Columnar Synchronization which Reflects Global Stimulus Properties* Nature **338**, pp. 334-337 (1989).

[3] Eckhorn R., Bauer R., Jordan W., Brosch M., Kruse W., Munk M., and Reitboeck H.J., *Coherent Oscillations: a Mechanism of Feature Linking in the Visual Cortex?*, Biol. Cyb. **60**, pp121-130 (1988).

[4] C. v.d.Malsburg *The Correlation Theory of Brain Function* Internal Report 81-2, Max-Planck-Institute for Biophysical Chemistry, Göttingen, F.R.G. (1981).

[5] Schuster H.G. (Ed.), Nonlinear Dynamics and Neuronal Networks, VCH Weinheim, Heidelberg (1991)

[6] Pawelzik K., Bauer H.-U., Geisel T. *Switching between predictable and unpredictable states in data from cat visual cortex*, talk at CNS San Francisco 1992, to appear in the CNS Proceedings.

[7] Gray C.M., Engel A.K., König P., Singer W., *Temporal Properties of Synchronous Oscillatory Interactions in Cat Striate Cortex*, in: Nonlinear Dynamics and Neuronal Networks, Ed. H.G. Schuster, VCH Weinheim, pp. 27-55 (1991)

[8] Deppisch J., Bauer H.-U., Schillen T., König P., Pawelzik K., Geisel T., *Stochastic and Oscillatory Burst Activities*, accepted for ICANN'92, Brighton, UK. (1992).

[9] Deppisch J., Bauer H.-U., Schillen T., König P., Pawelzik K., Geisel T., *Alternating Oscillatory and Stochastic States in a Network of Spiking Neurons*, submitted to Biol.Cyb. (1992).

[10] Rabiner, L.R., *A Tutorial on Hidden-Markov Models and Selected Applications in Speech Recognition* Proc. IEEE **77**, 2 pp. 257-286 (1989).

[11] Bauer H.-U., Deppisch J., Geisel T., Pawelzik K., in preparation.

[12] Bauer H.U., Pawelzik K., *Alternating Oscillatory and Stochastic Dynamics in a Model for a Neuronal Assembly*, Physica D, submitted.

[13] Schuster H.G., Koch C., *Burst Synchronization Without Frequency-Locking in a Completely Solvable Network Model*, in Moody J.E., Hanson S.J., Lippmann R.P. (Eds.), Neural Information Processing Systems 4, p. 117, Morgan Kauffmann (1992).